# *GenDeR*: A Generic Diversified Ranking Algorithm

**Jingrui He**
IBM T.J. Watson Research
Yorktown Heights, NY 10598
jingruhe@us.ibm.com

**Hanghang Tong**
IBM T.J. Watson Research
Yorktown Heights, NY 10598
htong@us.ibm.com

**Qiaozhu Mei**
University of Michigan
Ann Arbor, MI 48109
qmei@umich.edu

**Boleslaw K. Szymanski**
Rensselaer Polytechnic Institute
Troy, NY 12180
szymab@rpi.edu

## Abstract

*Diversified ranking* is a fundamental task in machine learning. It is broadly applicable in many real world problems, e.g., information retrieval, team assembling, product search, etc. In this paper, we consider a generic setting where we aim to diversify the top-k ranking list based on an *arbitrary* relevance function and an *arbitrary* similarity function among all the examples. We formulate it as an optimization problem and show that in general it is NP-hard. Then, we show that for a large volume of the parameter space, the proposed objective function enjoys the diminishing returns property, which enables us to design a scalable, greedy algorithm to find the $(1 - 1/e)$ *near-optimal* solution. Experimental results on real data sets demonstrate the effectiveness of the proposed algorithm.

## 1 Introduction

Many real applications can be reduced to a ranking problem. While traditional ranking tasks mainly focus on *relevance*, it has been widely recognized that *diversity* is another highly desirable property. It is not only a key factor to address the uncertainty and ambiguity in an information need, but also an effective way to cover the different aspects of the information need [14]. Take team assembling as an example. Given a task which typically requires a set of skills, we want to form a team of experts to perform that task. On one hand, each team member should have some *relevant* skills. On the other hand, the whole team should somehow be *diversified*, so that we can cover all the required skills for the task and different team members can benefit from each other's diversified, complementary knowledge and social capital. More recent research discovers that diversity plays a positive role in improving employees' performance within big organizations as well as their job retention rate in face of lay-off [21]; in improving the human-centric sensing results [15, 17]; in the decision of joining a new social media site (e.g., Facebook) [18], etc.

To date, many diversified ranking algorithms have been proposed. Early works mainly focus on text data [5, 23] where the goal is to improve the coverage of (sub-)topics in the retrieval result. In recently years, more attention has been paid to result diversification in web search [2, 20]. For example, if a query bears multiple meanings (such as the key word 'jaguar', which could refer to either cars or cats), we would like to have each meaning (e.g., 'cars' and 'cats' in the example of 'jaguar') covered by a subset of the top ranked web pages. Another recent trend is to diversify PageRank-type of algorithms for graph data [24, 11, 16]. It is worth pointing out that almost all the existing diversified ranking algorithms hinge on the *specific* choice of the relevance function and/or the similarity function. For example, in [2] and [20], both the relevance function and the similarity function implicitly depend on the categories/subtopics associated with the query and the documents; in [16], the

relevance function is obtained via personalized PageRank [8], and the similarity is measured based on the so-called 'Google matrix'; etc.

In this paper, we shift the problem to a more generic setting and ask: given an *arbitrary* relevance function wrt an implicit or explicit query, and an *arbitrary* similarity function among all the available examples, how can we diversify the resulting top-k ranking list? We address this problem from the optimization viewpoint. First, we propose an objective function that admits *any* non-negative relevance function and *any* non-negative, symmetric similarity function. It naturally captures both the relevance with regard to the query and the diversity of the ranking list, with a regularization parameter that balances between them. Then, we show that while such an optimization problem is NP-hard in general, for a large volume of the parameter space, the objective function exhibits the diminishing returns property, including submodularity, monotonicity, etc. Finally, we propose a scalable, greedy algorithm to find *provably near-optimal* solution.

The rest of the paper is organized as follows. We present our optimization formulation for diversified ranking in Section 2, followed by the analysis of its hardness and properties. Section 3 presents our greedy algorithm for solving the optimization problem. The performance of the proposed algorithm is evaluated in Section 4. In Section 5, we briefly review the related work. Finally, we conclude the paper in Section 6.

## 2 The Optimization Formulation

In this section, we present the optimization formulation for diversified ranking. We start by introducing the notation, and then present the objective function, followed by the analysis regarding its hardness and properties.

### 2.1 Notation

In this paper: we use normal lower-case letters to denote scalers or functions, bold-face lower-case letters to denote vectors, bold-face upper-case letters to denote matrices, and calligraphic upper-case letters to denote sets. To be specific, for a set $\mathcal{X}$ of $n$ examples $\{\boldsymbol{x}_1, \boldsymbol{x}_2, \ldots, \boldsymbol{x}_n\}$, let $\boldsymbol{S}$ denote the $n \times n$ similarity matrix, which is both symmetric and non-negative. In other words, $\boldsymbol{S}_{i,j} = \boldsymbol{S}_{j,i}$ and $\boldsymbol{S}_{i,j} \geq 0$, where $\boldsymbol{S}_{i,j}$ is the element of $\boldsymbol{S}$ in the $i^{\text{th}}$ row and the $j^{\text{th}}$ column $(i, j = 1, \ldots, n)$. For any ranking function $r(\cdot)$, which returns the non-negative relevance score for each example in $\mathcal{X}$ with respect to an implicit or explicit query, our goal is to find a subset $\mathcal{T}$ of $k$ examples, which are relevant to the query and diversified among themselves. Here the positive integer $k$ is the budget of the ranking list size, and the ranking function $r(\cdot)$ generates an $n \times 1$ vector $\boldsymbol{r}$, whose $i^{\text{th}}$ element $\boldsymbol{r}_i = r(\boldsymbol{x}_i)$. When we describe the objective function as well as the proposed optimization algorithm, it is convenient to introduce the following $n \times 1$ reference vector $\boldsymbol{q} = \boldsymbol{S} \cdot \boldsymbol{r}$. Intuitively, its $i^{\text{th}}$ element $\boldsymbol{q}_i$ measures the importance of $\boldsymbol{x}_i$. To be specific, if $\boldsymbol{x}_i$ is similar to many examples (high $\boldsymbol{S}_{i,j}$ $(j = 1, 2, ....,)$) that are relevant to the query (high $\boldsymbol{r}_j (j = 1, 2, ...)$, it is more important than the examples whose neighbors are not relevant. For example, if $\boldsymbol{x}_i$ is close to the center of a big cluster relevant to the query, the value of $\boldsymbol{q}_i$ is large.

### 2.2 Objective Function

With the above notation, our goal is to find a subset $\mathcal{T}$ of $k$ examples which are both relevant to the query and diversified among themselves. To this end, we propose the following optimization problem.

$$\arg\max_{|\mathcal{T}|=k} \quad \text{g}(\mathcal{T}) = w \sum_{i \in \mathcal{T}} \boldsymbol{q}_i \boldsymbol{r}_i - \sum_{i,j \in \mathcal{T}} \boldsymbol{r}_i \boldsymbol{S}_{i,j} \boldsymbol{r}_j \tag{1}$$

where $w$ is a positive regularization parameter that defines the trade-off between the two terms, and $\mathcal{T}$ consists of the indices of the $k$ examples that will be returned in the ranking list.

Intuitively, in the goodness function g($\mathcal{T}$), the first term measures the weighted overall relevance of $\mathcal{T}$ with respect to the query, and $\boldsymbol{q}_i$ is the weight for $\boldsymbol{x}_i$. It favors relevant examples from big clusters. In other words, if two examples are equally relevant to the query, one from a big cluster and the other isolated, by using the weighted relevance, we prefer the former. The second term measures

the similarity among the examples within $\mathcal{T}$. That is, it penalizes the selection of multiple relevant examples that are very similar to each other. By including this term in the objective function, we seek a set of examples which are relevant to the query, but also dissimilar to each other. For example, in the human-centric sensing [15, 17], due to the homophily in social networks, reports of two friends are likely correlated so that they are a lesser corroboration of events than reports of two socially unrelated witnesses.

## 2.3  The Hardness of Equation (1)

In the optimization problem in Equation (1), we want to find a subset $\mathcal{T}$ of $k$ examples that collectively maximize the goodness function $g(\mathcal{T})$. Unfortunately, by the following theorem, it is NP-hard to find the optimal solution.

**Theorem 2.1.** *The optimization problem in Equation* (1) *is NP-hard.*

*Proof.* We will prove this from the reduction of the Densest $k$-Subgraph (D$k$S) problem, which is known to be NP-hard [7].

To be specific, given an undirected graph $G(\mathcal{V}, \mathcal{E})$ with the connectivity matrix $\boldsymbol{W}$, where $\mathcal{V}$ is the set of vertices, and $\mathcal{E}$ is the set of edges. $\boldsymbol{W}$ is a $|\mathcal{V}| \times |\mathcal{V}|$ symmetric matrix with elements being 0 or 1. Let $|\mathcal{E}|$ be the total number of the edges in the graph. The D$k$S problem is defined in Equation (2).

$$\mathcal{Q} = \arg \max_{|\mathcal{Q}|=k} \sum_{i,j \in \mathcal{Q}} \boldsymbol{W}_{i,j} \tag{2}$$

Define another $|\mathcal{V}| \times |\mathcal{V}|$ matrix $\bar{\boldsymbol{W}}$ as: $\bar{\boldsymbol{W}}_{i,j} = 1 - \boldsymbol{W}_{i,j}$. It is easy to see that $\sum_{i,j \in \mathcal{Q}} \boldsymbol{W}_{i,j} = k^2 - \sum_{i,j \in \mathcal{Q}} \bar{\boldsymbol{W}}_{i,j}$. Therefore, Equation (2) is equivalent to

$$\mathcal{Q} = \arg \min_{|\mathcal{Q}|=k} \sum_{i,j \in \mathcal{Q}} \bar{\boldsymbol{W}}_{i,j} \tag{3}$$

Furthermore, notice that $\sum_{i,j=1}^{|\mathcal{V}|} \bar{\boldsymbol{W}}_{i,j} = |\mathcal{V}|^2 - |\mathcal{E}| = $ constant. Let $\mathcal{T} = \mathcal{V} \setminus \mathcal{Q}$, then Equation (3) is equivalent to

$$
\begin{aligned}
&\arg \max_{|\mathcal{Q}|=k} \sum_{i \in \mathcal{Q}, j \in \mathcal{T}} \bar{\boldsymbol{W}}_{i,j} + \sum_{i \in \mathcal{T}, j \in \mathcal{Q}} \bar{\boldsymbol{W}}_{i,j} + \sum_{i \in \mathcal{T}, j \in \mathcal{T}} \bar{\boldsymbol{W}}_{i,j} \\
= \ &\arg \max_{|\mathcal{T}|=|\mathcal{V}|-k} 2 \sum_{i \in \mathcal{Q}, j \in \mathcal{T}} \bar{\boldsymbol{W}}_{i,j} + \sum_{i,j \in \mathcal{T}} \bar{\boldsymbol{W}}_{i,j}
\end{aligned}
\tag{4}
$$

Next, we will show that Equation (4) can be viewed as an instance of the optimization problem in Equation (1) with the following setting: let the similarity function $\boldsymbol{S}$ be $\bar{\boldsymbol{W}}$, the ranking function $\boldsymbol{r}$ be $\mathbf{1}_{|\mathcal{V}| \times 1}$, the budget be $|\mathcal{V}| - k$, and the regularization parameter $w$ be 2. Under such settings, the objective function in Equation (1) becomes

$$
\begin{aligned}
g(\mathcal{T}) &= 2 \sum_{i \in \mathcal{T}} \boldsymbol{q}_i \boldsymbol{r}_i - \sum_{i,j \in \mathcal{T}} \boldsymbol{r}_i \bar{\boldsymbol{W}}_{i,j} \boldsymbol{r}_j \\
&= 2 \sum_{i \in \mathcal{T}} \sum_{j=1}^{|\mathcal{V}|} \boldsymbol{r}_i \bar{\boldsymbol{W}}_{ij} \boldsymbol{r}_j - \sum_{i,j \in \mathcal{T}} \boldsymbol{r}_i \bar{\boldsymbol{W}}_{i,j} \boldsymbol{r}_j \quad \text{(dfn. of } \mathbf{q}) \\
&= 2 \sum_{i \in \mathcal{Q}} \sum_{j \in \mathcal{T}} \boldsymbol{r}_i \bar{\boldsymbol{W}}_{ij} \boldsymbol{r}_j + \sum_{i,j \in \mathcal{T}} \boldsymbol{r}_i \bar{\boldsymbol{W}}_{i,j} \boldsymbol{r}_j \quad \text{(symmetry of } \bar{\mathbf{W}}) \\
&= 2 \sum_{i \in \mathcal{Q}} \sum_{j \in \mathcal{T}} \bar{\boldsymbol{W}}_{ij} + \sum_{i,j \in \mathcal{T}} \bar{\boldsymbol{W}}_{i,j} \quad \text{(dfn. of } \mathbf{r})
\end{aligned}
\tag{5}
$$

which is equivalent to the objective function in Equation (4). This completes the proof.  $\square$

## 2.4 Diminish Returns Property of $\mathbf{g}(\mathcal{T})$

Given that Equation (1) is NP-hard in general, we seek for a *provably near-optimal* solution instead in the next section. Here, let us first answer the following question: under what condition (e.g., in which range of the regularization parameter $w$), is it possible to find such a near-optimal solution for Equation (1)?

To this end, we present the so-called diminishing returns property of the goodness function $\mathbf{g}(\mathcal{T})$ defined in Equation (1), which is summarized in the following theorem. By Theorem 2.2, if we add more examples into an existing top-k ranking list, the goodness of the overall ranking list is non-decreasing (P2). However, the marginal benefit of adding additional examples into the ranking list decreases wrt the size of the existing ranking list (P1).

**Theorem 2.2. Diminish Returns Property of $\mathbf{g}(\mathcal{T})$.** *The goodness function $g(\mathcal{T})$ defined in Equation (1) has the following properties:*

> *(P1)* **submodularity**. *For any $w > 0$, the objective function $g(\mathcal{T})$ is submodular wrt $\mathcal{T}$;*
>
> *(P2)* **monotonicity**. *For any $w \geq 2$, The objective function $g(\mathcal{T})$ is monotonically non-decreasing wrt $\mathcal{T}$.*

*Proof.* We first prove (P1). For any $\mathcal{T}_1 \subset \mathcal{T}_2$ and any given example $\boldsymbol{x} \notin \mathcal{T}_2$, we have

$$
\begin{aligned}
\mathbf{g}(\mathcal{T}_1 \cup \boldsymbol{x}) - \mathbf{g}(\mathcal{T}_1) &= (w \sum_{i \in \mathcal{T}_1 \cup \boldsymbol{x}} \boldsymbol{q}_i \boldsymbol{r}_i - \sum_{i,j \in \mathcal{T}_1 \cup \boldsymbol{x}} \boldsymbol{r}_i \boldsymbol{S}_{i,j} \boldsymbol{r}_j) - (w \sum_{i \in \mathcal{T}_1} \boldsymbol{q}_i \boldsymbol{r}_i - \sum_{i,j \in \mathcal{T}_1} \boldsymbol{r}_i \boldsymbol{S}_{i,j} \boldsymbol{r}_j) \\
&= w \boldsymbol{q}_{\boldsymbol{x}} \boldsymbol{r}_{\boldsymbol{x}} - (\sum_{i \in \mathcal{T}_1} \boldsymbol{r}_i \boldsymbol{S}_{i,\boldsymbol{x}} \boldsymbol{r}_{\boldsymbol{x}} + \sum_{j \in \mathcal{T}_1} \boldsymbol{r}_{\boldsymbol{x}} \boldsymbol{S}_{\boldsymbol{x},j} \boldsymbol{r}_j + \boldsymbol{r}_{\boldsymbol{x}} \boldsymbol{S}_{\boldsymbol{x},\boldsymbol{x}} \boldsymbol{r}_{\boldsymbol{x}}) \\
&= w \boldsymbol{q}_{\boldsymbol{x}} \boldsymbol{r}_{\boldsymbol{x}} - \boldsymbol{S}_{\boldsymbol{x},\boldsymbol{x}} \boldsymbol{r}_{\boldsymbol{x}}^2 - 2 \boldsymbol{r}_{\boldsymbol{x}} \sum_{j \in \mathcal{T}_1} \boldsymbol{S}_{\boldsymbol{x},j} \boldsymbol{r}_j \qquad (6)
\end{aligned}
$$

Similarly, we have $\mathbf{g}(\mathcal{T}_2 \cup \boldsymbol{x}) - \mathbf{g}(\mathcal{T}_2) = w \boldsymbol{q}_{\boldsymbol{x}} \boldsymbol{r}_{\boldsymbol{x}} - \boldsymbol{S}_{\boldsymbol{x},\boldsymbol{x}} \boldsymbol{r}_{\boldsymbol{x}}^2 - 2 \boldsymbol{r}_{\boldsymbol{x}} \sum_{j \in \mathcal{T}_2} \boldsymbol{S}_{\boldsymbol{x},j} \boldsymbol{r}_j$.

Therefore, we have

$$
\begin{aligned}
(\mathbf{g}(\mathcal{T}_1 \cup \boldsymbol{x}) - \mathbf{g}(\mathcal{T}_1)) - (\mathbf{g}(\mathcal{T}_2 \cup \boldsymbol{x}) - \mathbf{g}(\mathcal{T}_2)) &= 2 \boldsymbol{r}_{\boldsymbol{x}} \sum_{j \in \mathcal{T}_2} \boldsymbol{S}_{\boldsymbol{x},j} \boldsymbol{r}_j - 2 \boldsymbol{r}_{\boldsymbol{x}} \sum_{j \in \mathcal{T}_1} \boldsymbol{S}_{\boldsymbol{x},j} \boldsymbol{r}_j \\
&= 2 \boldsymbol{r}_{\boldsymbol{x}} \sum_{j \in \mathcal{T}_2 \setminus \mathcal{T}_1} \boldsymbol{S}_{\boldsymbol{x},j} \boldsymbol{r}_j \geq 0 \qquad (7)
\end{aligned}
$$

which completes the proof of (P1).

Next, we prove (P2). Given any $\mathcal{T}_1 \cap \mathcal{T}_2 = \Phi$, where $\Phi$ is the empty set, with $w \geq 2$, we have

$$
\begin{aligned}
\mathbf{g}(\mathcal{T}_2 \cup \mathcal{T}_1) - \mathbf{g}(\mathcal{T}_2) &= w \sum_{i \in \mathcal{T}_1} \boldsymbol{q}_i \boldsymbol{r}_i - (\sum_{i \in \mathcal{T}_1, j \in \mathcal{T}_2} \boldsymbol{r}_i \boldsymbol{S}_{i,j} \boldsymbol{r}_j + \sum_{i \in \mathcal{T}_2, j \in \mathcal{T}_1} \boldsymbol{r}_i \boldsymbol{S}_{i,j} \boldsymbol{r}_j + \sum_{i,j \in \mathcal{T}_1} \boldsymbol{r}_i \boldsymbol{S}_{i,j} \boldsymbol{r}_j) \\
&= w \sum_{i \in \mathcal{T}_1} \boldsymbol{r}_i \sum_{j=1}^{n} \boldsymbol{S}_{i,j} \boldsymbol{r}_j - (2 \sum_{i \in \mathcal{T}_1, j \in \mathcal{T}_2} \boldsymbol{r}_i \boldsymbol{S}_{i,j} \boldsymbol{r}_j + \sum_{i,j \in \mathcal{T}_1} \boldsymbol{r}_i \boldsymbol{S}_{i,j} \boldsymbol{r}_j) \\
&\geq 2 \sum_{i \in \mathcal{T}_1} \boldsymbol{r}_i \sum_{j=1}^{n} \boldsymbol{S}_{i,j} \boldsymbol{r}_j - 2(\sum_{i \in \mathcal{T}_1, j \in \mathcal{T}_2} \boldsymbol{r}_i \boldsymbol{S}_{i,j} \boldsymbol{r}_j + \sum_{i,j \in \mathcal{T}_1} \boldsymbol{r}_i \boldsymbol{S}_{i,j} \boldsymbol{r}_j) \\
&= 2 \sum_{i \in \mathcal{T}_1} \boldsymbol{r}_i (\sum_{j=1}^{n} \boldsymbol{S}_{i,j} \boldsymbol{r}_j - \sum_{j \in \mathcal{T}_1 \cup \mathcal{T}_2} \boldsymbol{S}_{i,j} \boldsymbol{r}_j) \\
&= 2 \sum_{i \in \mathcal{T}_1} \boldsymbol{r}_i \sum_{j \notin \mathcal{T}_1 \cup \mathcal{T}_2} \boldsymbol{S}_{i,j} \boldsymbol{r}_j \geq 0 \qquad (8)
\end{aligned}
$$

which completes the proof of (P2). $\qquad \square$

# 3 The Optimization Algorithm

In this section, we present our algorithm GenDeR for solving Equation (1), and analyze its performance with respect to its near-optimality and complexity.

## 3.1 Algorithm Description

Based on the diminishing returns property of the goodness function $g(\mathcal{T})$, we propose the following greedy algorithm to find a diversified top-k ranking list. In Alg. 1, after we calculate the reference vector $q$ (Step 1) and initialize the ranking list $\mathcal{T}$ (Step 2), we try to expand the ranking list $\mathcal{T}$ one-by-one (Step 4-8). At each iteration, we add one more example with the highest score $s_i$ into the current ranking list $\mathcal{T}$ (Step 5). Each time we expand the current ranking list, we update the score vector $s$ based on the newly added example $i$ (Step 7). Notice that in Alg. 1, '$\otimes$' means the element-wise multiplication, and $\mathrm{diag}(S)$ returns an $n \times 1$ vector with the corresponding elements being the diagonal elements in the similarity matrix $S$.

---

**Algorithm 1** GenDeR

**Input:** The similarity matrix $S_{n \times n}$, the relevance vector $r_{n \times 1}$, the weight $w \geq 2$, and the budget $k$;

**Output:** A subset $\mathcal{T}$ of $k$ nodes.
1: Compute the reference vector $q$: $q = Sr$;
2: Initialize $\mathcal{T}$ as an empty set;
3: Initialize the score vector $s = w \times (q \otimes r) - \mathrm{diag}(S) \otimes r \otimes r$;
4: **for** iter $= 1 : k$ **do**
5:     Find $i = \mathrm{argmax}_j(s_j | j = 1, ..., n; j \notin \mathcal{T})$;
6:     Add $i$ to $\mathcal{T}$;
7:     Update the score vector $s \leftarrow s - 2r_i S_{:,i} \otimes r$
8: **end for**
9: Return the subset $\mathcal{T}$ as the ranking list (earlier selected examples ranked higher).

---

## 3.2 Algorithm Analysis

The accuracy of the proposed GenDeR is summarized in Lemma 3.1, which says that for a large volume of the parameter space (i.e., $w \geq 2$), GenDeR leads to a $(1 - 1/e)$ near-optimal solution.

**Lemma 3.1. Near-Optimality of** GenDeR. *Let $\mathcal{T}$ be the subset found by* GenDeR, *$|\mathcal{T}| = k$, and $\mathcal{T}^* = argmax_{|\mathcal{T}|=k} g(\mathcal{T})$. We have that $g(\mathcal{T}) \geq (1 - 1/e)g(\mathcal{T}^*)$, where $e$ is the base of the natural logarithm.*

*Proof.* The key of the proof is to verify that for any example $x_j \notin \mathcal{T}$, $s_j = g(\mathcal{T} \cup x_j) - g(\mathcal{T})$, where $s$ is the score vector we calculate in Step 3 or update in Step 7, and $\mathcal{T}$ is the initial empty ranking list or the current ranking list in Step 6. The remaining part of the proof directly follows the diminishing returns property of the goodness function in Theorem 2.2, together with the fact that $g(\Phi) = 0$ [12]. We omit the detailed proof for brevity. □

The complexity of the proposed GenDeR is summarized in Lemma 3.2. Notice that the quadratic term in the time complexity comes from the matrix-vector multiplication in Step 1 (i.e., $q = Sr$); and the quadratic term in the space complexity is the cost to store the similarity matrix $S$. If the similarity matrix $S$ is sparse, say we have $m$ non-zero elements in $S$, we can reduce the time complexity to $O(m + nk)$; and reduce the space complexity to $O(m + n + k)$.

**Lemma 3.2. Complexity of** GenDeR. *The time complexity of* GenDeR *is $O(n^2 + nk)$; the space complexity of* GenDeR *is $O(n^2 + k)$.*

*Proof.* Omitted for Brevity. □

# 4 Experimental Results

We compare the proposed `GenDeR` with several most recent diversified ranking algorithms, including DivRank based on reinforced random walks [11] (referred to as 'DR'), GCD via resistive graph centers [6] (referred to as 'GCD') and manifold ranking with stop points [25] (referred to as 'MF'). As all these methods aim to improve the diversity of PageRank-type of algorithms, we also present the results by PageRank [13] itself as the baseline. We use two real data sets, including an IMDB actor professional network and an academic citation data set. In [11, 6], the authors provide detailed experimental comparisons with some earlier methods (e.g., [24, 23, 5], etc) on the same data sets. We omit the results by these methods for clarity.

## 4.1 Results on Actor Professional Network

The actor professional network is constructed from the Internet Movie Database (IMDB)[1], where the nodes are the actors/actresses and the edges are the numbers of the co-stared movies between two actors/actresses. For the inputs of `GenDeR`, we use the adjacency matrix of the co-stared network as the similarity function $S$; and the ranking results by 'DR' as the relevance vector $r$. Given a top-k ranking list, we use the density of the induced subgraph of $S$ by the $k$ nodes as the reverse measure of the diversity (lower density means higher diversity). We also measure the diversity of the ranking list by the so-called 'country coverage' as well as 'movie coverage' (higher coverage means higher diversity), which are defined in [24]. Notice that for a good top-k diversified ranking list, it often requires the balance between the diversity and the relevance in order to fulfill the user's information need. Therefore, we also present the relevance score (measured by PageRank) captured by the entire top-k ranking list. In this application, such a relevance score measures the overall prestige of the actors/actresses in the ranking list. Overall, we have 3,452 actors/actresses, 23,460 edges, 1,027 movies and 47 countries.

The results are presented in Fig. 1. First, let us compare `GenDeR` with the baseline method 'PageRank'. From Fig. 1(d), we can see that our `GenDeR` is as good as 'PageRank' in terms of capturing the relevance of the entire top-k ranking list (notice that the two curves almost overlap with each other). On the other hand, `GenDeR` outperforms 'PageRank' in terms of the diversity by all the three measures (Fig. 1(a-c)). Since `GenDeR` uses the ranking results by 'DR' as its input, 'DR' can be viewed as another baseline method. The two methods perform similarly in terms of density (Fig. 1(c)). Regarding all the remaining measures, our `GenDeR` is always better than 'DR'. For example, when $k \geq 300$, `GenDeR` returns both higher 'country-coverage' (Fig. 1(a)) and higher 'movie-coverage' (Fig. 1(b)). In the entire range of the budget $k$ (Fig. 1(d)), our `GenDeR` captures higher relevance scores than 'DR', indicating the actors/actresses in our ranking list might be more prestigious than those by 'DR'. Based on these results, we conclude that our `GenDeR` indeed improves 'DR' in terms of both diversity and relevance. The most competitive method is 'MF'. We can see that `GenDeR` and 'MF' perform similarly in terms of both density (Fig. 1(c)) and 'movie coverage' (Fig. 1(b)). In terms of 'country coverage' (Fig. 1(a)), 'MF' performs slightly better than our `GenDeR` when $300 \leq k \leq 400$; and for the other values of $k$, the two methods mix with each other. However, in terms of relevance (Fig. 1(d)), our `GenDeR` is much better than 'MF'. Therefore, we conclude that 'MF' performs comparably with or slightly better than our `GenDeR` in terms of diversity, at the cost of sacrificing the relevance of the entire ranking list. As for 'GCD', although it leads to the lowest density, it performs poorly in terms of balancing between the diversity and the relevance (Fig. 1(d)), as well as the coverage of countries/movies (Fig. 1(a-b)).

## 4.2 Results on Academic Citation Networks

This data set is from ACL Anthology Network[2]. It consists of a paper citation network and a researcher citation network. Here, the nodes are papers or researchers; and the edges indicate the citation relationship. Overall, we have 11,609 papers and 54,208 edges in the paper citation network; 9,641 researchers and 229,719 edges in the researcher citation network. For the inputs of `GenDeR`, we use the symmetrized adjacency matrix as the similarity function $S$; and the ranking results by 'DR' as the relevance vector $r$. We use the same measure as in [11] (referred to as 'coverage'), which is the total number of unique papers/researchers that cite the top-k papers/researchers in the ranking list. As pointed out in [11], the 'coverage' might provide a better measure of the overall quality of the top-k ranking list than those traditional measures (e.g., h-index) as they ignore the diversity of the ranking list. The results are presented in Fig. 2. We can see that the proposed `GenDeR`

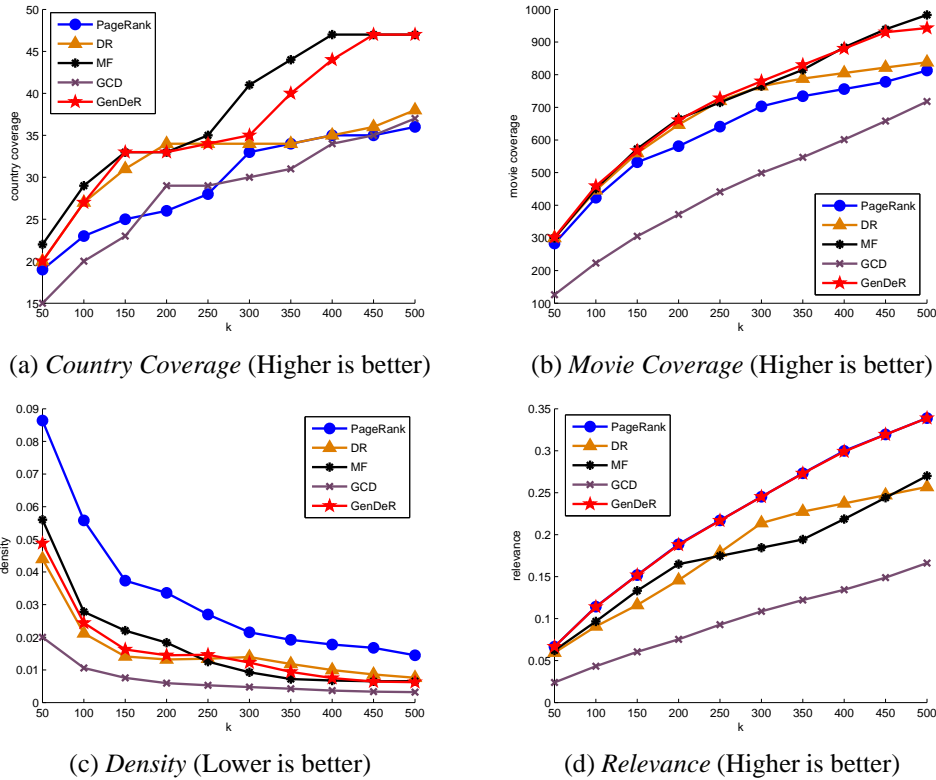

(a) *Country Coverage* (Higher is better)

(b) *Movie Coverage* (Higher is better)

(c) *Density* (Lower is better)

(d) *Relevance* (Higher is better)

Figure 1: The evaluations on actor professional network. (a-c) are different diversity measures and (d) measures the relevance of the entire ranking list.

performs better than all the alternative choices. For example, with $k = 50$, GenDeR improves the 'coverage' of the next best method by 416 and 157 on the two citation networks, respectively.

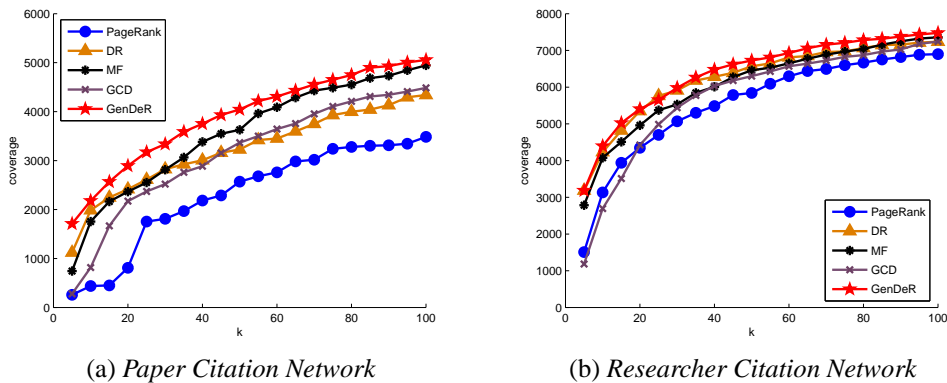

(a) *Paper Citation Network*

(b) *Researcher Citation Network*

Figure 2: The evaluations on academic citation networks. Higher is better.

## 5   Related Work

Carbonell et al [5] are among the first to study diversified ranking in the context of text retrieval and summarization. To this end, they propose to use the Maximal Marginal Relevance (MMR) criterion to reduce redundancy while maintaining query relevance, which is a linear combination of relevance and novelty. In [23], Zhai et al address this problem from a different perspective by explicitly modeling the subtopics associated with a query, and proposing a framework to evaluate subtopic retrieval. Recently, researchers leverage external information sources to help with diversified ranking. For example, in [2], Agrawal et al maximize the probability that the average user finds at least one useful

result within the top ranked results with the help of a taxonomy available through Open Directory Project (ODP); in [4], Capannini et al mine the query log to find specializations of a given query, and use the search results of the specializations to help evaluate the set of top ranked documents; in [20], Welch et al model the expected number of hits based on the number of relevant documents a user will visit, user intent in terms of the probability distribution over subtopics, and document categorization, which are obtained from the query logs, WordNet or Wikipedia.

With the prevalence of graph data, such as social networks, author/paper citation networks, actor professional networks, etc, researchers have started to study the problem of diversified ranking in the presence of relationships among the examples. For instance, in [24], Zhu et al propose the GRASSHOPPER algorithm by constructing random walks on the input graph, and iteratively turning the ranked nodes into absorbing states. In [11], Mei et al propose the DivRank algorithm based on a reinforced random walk defined on the input graph, which automatically balances the prestige and the diversity among the top ranked nodes due to the fact that adjacent nodes compete for their ranking scores. In [16], Tong et al propose a scalable algorithm to find the near-optimal solution to diversify the top-k ranking list for PageRank. Due to the asymmetry in their formulation, it remains unclear if the optimization problem in [16] is NP-hard. On a higher level, the method in [16] can be roughly viewed as an instantiation of our proposed formulation with the specific choices in the optimization problem (e.g, the relevance function, the similarity function, the regularization parameter, etc). In [25], Zhu et al leverage the stopping points in the manifold ranking algorithms to diversify the results. All these works aim to diversify the results of one *specific* type of ranking function (i.e., PageRank and its variants).

Learning to rank [10, 1, 3] and metric learning [19, 22, 9] have been two very active areas in the recent years. Most of these methods require some additional information (e.g., label, partial ordering, etc) for training. They are often tailored for other purposes (e.g., improving the F-score in the ranking task, improving the classification accuracy in metric learning, etc) without the consideration of diversity. Nonetheless, thanks to the generality of our formulation, the learned ranking functions and metric functions from most of these works can be naturally admitted into our optimization objective function. In other words, our formulation brings the possibility to take advantage of these existing research results in the diversified ranking setting.

*Remarks.* While generality is one of the major contributions of this paper, we do not disregard the value of the domain-specific knowledge. The generality of our method is orthogonal to domain-specific knowledge. For example, such knowledge can be reflected in the (learnt) ranking function and/or the (learnt) similarity function, which can in turn serve as the input of our method.

## 6  Conclusion

In this paper, we study the problem of diversified ranking. The key feature of our formulation lies in its *generality*: it admits *any* non-negative relevance function and *any* non-negative, symmetric similarity function as input, and outputs a top-k ranking list that enjoys both relevance and diversity. Furthermore, we identify the regularization parameter space where our problem can be solved near-optimally; and we analyze the *hardness* of the problem, the *optimality* as well as the *complexity* of the proposed algorithm. Finally, we conduct experiments on several real data sets to demonstrate the effectiveness of this algorithm. Future work includes extending our formulation to the on-line, dynamic setting.

## 7  Acknowledgement

Research was sponsored by the Army Research Laboratory and was accomplished under Cooperative Agreement Number W911NF-09-2-0053. This work was in part supported by the National Science Foundation under grant numbers IIS-1054199 and CCF-1048168; and by DAPRA under SMISC Program Agreement No. W911NF-12-C-0028. The views and conclusions contained in this document are those of the authors and should not be interpreted as representing the official policies, either expressed or implied, of the Army Research Laboratory, the National Science Foundation, or the U.S. Government. The U.S. Government is authorized to reproduce and distribute reprints for Government purposes notwithstanding any copyright notation here on.

## Footnotes

[1]http://www.imdb.com/

[2]http://www.aclweb.org/anthology-new/

## References

[1] A. Agarwal and S. Chakrabarti. Learning random walks to rank nodes in graphs. In *ICML*, pages 9–16, 2007.

[2] R. Agrawal, S. Gollapudi, A. Halverson, and S. Ieong. Diversifying search results. In *WSDM*, pages 5–14, 2009.

[3] C. J. C. Burges, K. M. Svore, P. N. Bennett, A. Pastusiak, and Q. Wu. Learning to rank using an ensemble of lambda-gradient models. *Journal of Machine Learning Research - Proceedings Track*, 14:25–35, 2011.

[4] G. Capannini, F. M. Nardini, R. Perego, and F. Silvestri. Efficient diversification of search results using query logs. In *WWW (Companion Volume)*, pages 17–18, 2011.

[5] J. G. Carbonell and J. Goldstein. The use of mmr, diversity-based reranking for reordering documents and producing summaries. In *SIGIR*, pages 335–336, 1998.

[6] A. Dubey, S. Chakrabarti, and C. Bhattacharyya. Diversity in ranking via resistive graph centers. In *KDD*, pages 78–86, 2011.

[7] U. Feige, G. Kortsarz, and D. Peleg. The dense k-subgraph problem. *Algorithmica*, 29, 1999.

[8] T. H. Haveliwala. Topic-sensitive pagerank: A context-sensitive ranking algorithm for web search. *IEEE Trans. Knowl. Data Eng.*, 15(4):784–796, 2003.

[9] P. Jain, B. Kulis, and I. S. Dhillon. Inductive regularized learning of kernel functions. In *NIPS*, pages 946–954, 2010.

[10] T.-Y. Liu. Learning to rank for information retrieval. In *SIGIR*, page 904, 2010.

[11] Q. Mei, J. Guo, and D. R. Radev. Divrank: the interplay of prestige and diversity in information networks. In *KDD*, pages 1009–1018, 2010.

[12] G. L. Nemhauser, L. A. Wolsey, and M. L. Fisher. An analysis of approximations for maximizing submodular set functionsłi. *MATHEMATICAL PROGRAMMING*, (1):265–294, 1973.

[13] L. Page, S. Brin, R. Motwani, and T. Winograd. The PageRank citation ranking: Bringing order to the web. Technical report, Stanford Digital Library Technologies Project, 1998. Paper SIDL-WP-1999-0120 (version of 11/11/1999).

[14] F. Radlinski, P. N. Bennett, B. Carterette, and T. Joachims. Redundancy, diversity and interdependent document relevance. *SIGIR Forum*, 43(2):46–52, 2009.

[15] M. Srivastava, T. Abdelzaher, and B. Szymanski. Human-centric sensing. *Phil. Trans. R. Soc. 370 ser. A(1958)*, pages 176–197, 2012.

[16] H. Tong, J. He, Z. Wen, R. Konuru, and C.-Y. Lin. Diversified ranking on large graphs: an optimization viewpoint. In *KDD*, pages 1028–1036, 2011.

[17] M. Y. S. Uddin, M. T. A. Amin, H. Le, T. Abdelzaher, B. Szymanski, and T. Nguyen. On diversifying source selection in social sensing. In *INSS*, 2012.

[18] J. Ugander, L. Backstrom, C. Marlow, and J. Kleinberg. Structural diversity in social contagion. *PNAS*, 109(16):596–5966, 2012.

[19] J. Wang, H. Do, A. Woznica, and A. Kalousis. Metric learning with multiple kernels. In *NIPS*, pages 1170–1178, 2011.

[20] M. J. Welch, J. Cho, and C. Olston. Search result diversity for informational queries. In *WWW*, pages 237–246, 2011.

[21] L. Wu. Social network effects on performance and layoffs: Evidence from the adoption of a social networking tool. *Job Market Paper*, 2011.

[22] E. P. Xing, A. Y. Ng, M. I. Jordan, and S. J. Russell. Distance metric learning with application to clustering with side-information. In *NIPS*, pages 505–512, 2002.

[23] C. Zhai, W. W. Cohen, and J. D. Lafferty. Beyond independent relevance: methods and evaluation metrics for subtopic retrieval. In *SIGIR*, pages 10–17, 2003.

[24] X. Zhu, A. B. Goldberg, J. V. Gael, and D. Andrzejewski. Improving diversity in ranking using absorbing random walks. In *HLT-NAACL*, pages 97–104, 2007.

[25] X. Zhu, J. Guo, X. Cheng, P. Du, and H. Shen. A unified framework for recommending diverse and relevant queries. In *WWW*, pages 37–46, 2011.

